# Cue Integration for Figure/Ground Labeling

**Xiaofeng Ren, Charless C. Fowlkes and Jitendra Malik**
Computer Science Division, University of California, Berkeley, CA 94720
{xren,fowlkes,malik}@cs.berkeley.edu

## Abstract

We present a model of edge and region grouping using a conditional random field built over a scale-invariant representation of images to integrate multiple cues. Our model includes potentials that capture low-level similarity, mid-level curvilinear continuity and high-level object shape. Maximum likelihood parameters for the model are learned from human labeled groundtruth on a large collection of horse images using belief propagation. Using held out test data, we quantify the information gained by incorporating generic mid-level cues and high-level shape.

## 1 Introduction

Figure/ground organization, the binding of contours to surfaces, is a classical problem in vision. In the 1920s, Edgar Rubin pointed to several generic properties, such as closure, which governed the perception of figure/ground. However, it is clear that in the context of natural scenes, such processing must be closely intertwined with many low- and mid-level grouping cues as well as a priori object knowledge [10].

In this paper, we study a simplified task of figure/ground labeling in which the goal is to label every pixel as belonging to either a figural object or background. Our goal is to understand the role of different cues in this process, including low-level cues, such as edge contrast and texture similarity; mid-level cues, such as curvilinear continuity; and high-level cues, such as characteristic shape or texture of the object. We develop a conditional random field model [7] over edges, regions and objects to integrate these cues. We train the model from human-marked groundtruth labels and quantify the relative contributions of each cue on a large collection of horse images[2].

In computer vision, the work of Geman and Geman [3] inspired a whole subfield of work on Markov Random Fields in relation to segmentation and denoising. More recently, *Conditional Random Fields* (CRF) have been applied to low-level segmentation [6, 12, 4] and have shown performance superior to traditional MRFs. However, most of the existing MRF/CRF models focus on pixel-level labeling, requiring inferences over millions of pixels. Being tied to the pixel resolution, they are also unable to deal with scale change or explicitly capture mid-level cues such as junctions. Our approach overcomes these difficulties by utilizing a scale-invariant representation of image contours and regions where each variable in our model can correspond to hundreds of pixels. It is also quite straightforward to design potentials which capture complicated relationships between these mid-level tokens in a transparent way.

Interest in combining object knowledge with segmentation has grown quickly over the

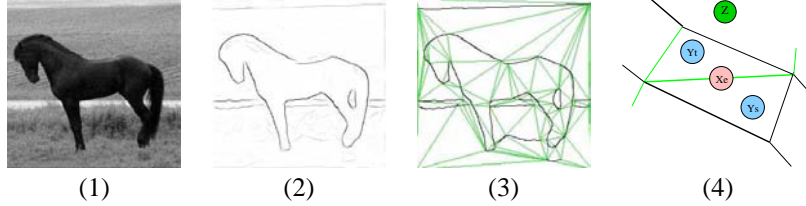

(1)        (2)        (3)        (4)

Figure 1: A scale-invariant representation of images: Given the input (1), we estimate the local probability of boundary $Pb$ based on gradients (2). We then build a piecewise linear approximation of the edge map and complete it with *Constrained Delaunay Triangulation* (CDT). The black edges in (3) are gradient edges detected in (2); the green edges are potential completions generated by CDT. (4) We perform inference in a probabilistic model built on top of this representation and extract marginal distributions on edges $X$, triangular regions $Y$ and object pose $Z$.

last few years [2, 16, 14]. Our probabilistic approach is similar in spirit to [14] however we focus on learning parameters of a discriminative model and quantify our performance on test data. Compared to previous techniques which rely heavily on top-down template matching [2, 5], our approach has three major advantages: (1) We are able to use mid-level grouping cues including junctions and continuity. Our results show these cues make quantitatively significant contributions. (2) We combine cues in a probabilistic framework where the relative weighting of cues is learned from training data resulting in weights that are easy to interpret. (3) The role of different cues can be easily studied by "surgically removing" them refitting the remaining parameters.

## 2    A conditional random field for figure/ground labeling

Figure 1 provides an overview of our technique for building a discrete, scale-independent representation of image boundaries from a low-level detector. First we compute an edge map using the boundary detector of [9] which utilizes both brightness and texture contrast to estimate the probability of boundary, $Pb$ at each pixel. Next we use Canny's hysteresis thresholding to trace the $Pb$ boundaries and then recursively split the boundaries using angles, a scale-invariant measure, until each segment is approximately linear. Finally we utilize the *Constrained Delaunay Triangulation* [13] to complete the piecewise linear approximations. CDT often completes gaps in object boundaries where local gradient information is absent. More details about this construction can be found in [11].

Let $G$ be the resulting CDT graph. The edges and triangles in $G$ are natural entities for figure/ground labeling. We introduce the following random variables:

- Edges: $X_e$ is 1 if edge $e$ in the CDT is a true boundary and 0 otherwise.

- Regions: $Y_t$ is 1 if triangle $t$ corresponds to figure and 0 otherwise.

- Pose: $Z$ encodes the figural object's pose in the scene. We use a very simple $Z$ which considers a discrete configuration space given by a grid of 25 possible image locations. $Z$ is easily augmented to include an indicator of object category or aspect as well as location.

We now describe a conditional random field model on $\{X, Y, Z\}$ used to integrate multiple grouping cues. The model takes the form of a log-linear combination of features which are functions of variables and image measurements. We consider $Z$ a latent variable which is

marginalized out by assuming a uniform distribution over aspects and locations.

$$P(X,Y|Z,I,\Theta) = \frac{1}{\mathcal{Z}(I,\Theta)} e^{-E(X,Y|Z,I,\Theta)}$$

where the energy $E$ of a configuration is linear in the parameters $\Theta = \{\alpha, \vec{\beta}, \vec{\delta}, \gamma, \vec{\eta}, \kappa, \vec{\nu}\}$ and given by

$$E = -\alpha \sum_e L_1(X_e|I) - \vec{\beta} \cdot \sum_{\langle s,t \rangle} \vec{L}_2(Y_s, Y_t|I) - \vec{\delta} \cdot \sum_V \vec{M}_1(X_V|I)$$

$$-\gamma \sum_{\langle s,t \rangle} M_2(Y_s, Y_t, X_e) - \vec{\eta} \cdot \sum_t \vec{H}_1(Y_t|I) - \kappa \sum_t H_2(Y_t|Z,I) - \vec{\nu} \cdot \sum_e \vec{H}_3(X_e|Z,I)$$

The table below gives a summary of each potential. The next section fills in details.

| Similarity | Edge energy along $e$ | $L_1(X_e|I)$ |
|---|---|---|
| | Brightness/Texture similarity between $s$ and $t$ | $L_2(Y_s, Y_t|I)$ |
| Continuity | Collinearity and junction frequency at vertex $V$ | $M_1(X_V|I)$ |
| Closure | Consistency of edge and adjoining regions | $M_2(Y_s, Y_t, X_e)$ |
| | Similarity of region $t$ to exemplar texture | $H_1(Y_t|I)$ |
| Familiarity | Compatibility of region shape with pose | $H_2(Y_t|Z,I)$ |
| | Compatibility of local edge shape with pose | $H_3(X_e|Z,I)$ |

## 3 Cues for figure/ground labeling

### 3.1 Low-level Cues: Similarity of Brightness and Texture

To capture the locally measured edge contrast, we assign a singleton edge potential whose energy is

$$L_1(X_e|I) = \log(Pb_e)X_e$$

where $Pb_e$ is the average $Pb$ recorded over the pixels corresponding to edge $e$.

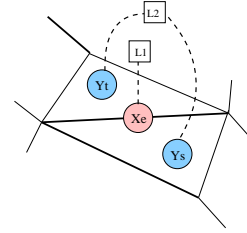

Since the triangular regions have larger support than the local edge detector, we also include a pairwise, region-based similarity cue, computed as

$$\vec{\beta} \cdot \vec{L}_2(Y_s, Y_t|I) = (\beta_B \log(f(|I_s - I_t|)) + \beta_T \log(g(\chi^2(h_s, h_t))))\mathbf{1}_{\{Y_s=Y_t\}}$$

where $f$ predicts the likelihood of $s$ and $t$ belonging to the same group given the difference of average image brightness and $g$ makes a similar prediction based on the $\chi^2$ difference between histograms of vector quantized filter responses (referred to as textons [8]) which describe the texture in the two regions.

### 3.2 Mid-level Cues: Curvilinear Continuity and Closure

There are two types of edges in the CDT graph, gradient-edges (detected by $Pb$) and completed-edges (filled in by the triangulation). Since true boundaries are more commonly marked by a gradient, we keep track of these two types of edges separately when modeling junctions. To capture continuity and the frequency of different junction types, we assign energy:

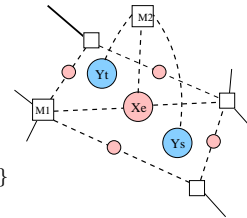

$$\vec{\delta} \cdot \vec{M}_1(X_V|I) = \sum_{i,j} \delta_{i,j} \mathbf{1}_{\{\deg_g(V)=i, \deg_c(V)=j\}}$$

$$+ \delta_C \mathbf{1}_{\{\deg_g(V)+\deg_c(V)=2\}} \log(h(\theta))$$

where $X_V = \{X_{e_1}, X_{e_2}, \ldots\}$ is the set of edge variables incident on $V$, $\deg_g(V)$ is the number of gradient-edges at vertex $V$ for which $X_e = 1$. Similarly $\deg_c(V)$ is the number of completed-edges that are "turned on". When the total degree of a vertex is 2, $\delta_C$ weights the continuity of the two edges. $h$ is the output of a logistic function fit to $|\theta|$ and the probability of continuation. It is smooth and symmetric around $\theta = 0$ and falls of as $\theta \to \pi$. If the angle between the two edges is close to 0, they form a good continuation, $f(\theta)$ is large, and they are more likely to both be turned on.

In order to assert the duality between segments and boundaries, we use a compatibility term

$$M_2(Y_s, Y_t, X_e) = \mathbf{1}_{\{Y_s = Y_t, X_e = 0\}} + \mathbf{1}_{\{Y_s \neq Y_t, X_e = 1\}}$$

which simply counts when the label of $s$ and $t$ is consistent with that of $e$.

### 3.3  High-level Cues: Familiarity of Shape and Texture

We are interested in encoding high-level knowledge about object categories. In this paper we experiment with a single object category, horses, but we believe our high-level cues will scale to multiple objects in a natural way.

We compute texton histograms $h_t$ for each triangular region (as in $L_1$). From the set of training images, we use k-medoids to find 10 representative histograms $\{h_1^F, \ldots, h_{10}^F\}$ for the collection of segments labeled as figure and 10 histograms $\{h_l^G, \ldots, h_{l0}^G\}$ for the set of background segments. Each segment in a test image is compared to the set of exemplar histograms using the $\chi^2$ histogram difference. We use the energy term

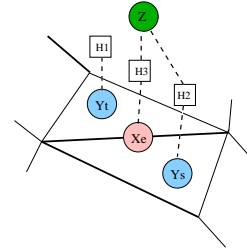

$$H_1(Y_t|I) = \log\left(\frac{\min_i \chi^2(h_t, h_i^F)}{\min_i \chi^2(h_t, h_i^G)}\right) Y_t$$

to capture the cue of texture familiarity.

We describe the global shape of the object using a template $T(x, y)$ generated by averaging the groundtruth object segmentation masks. This yields a silhouette with quite fuzzy boundaries due to articulations and scale variation. Figure 3.3(a) shows the template extracted from our training data. Let $O(Z, t)$ be the normalized overlap between template centered at $Z = (x_0, y_0)$ with the triangular region corresponding to $Y_t$. This is computed as the integral of $T(x, y)$ over the triangle $t$ divided by the area of $t$. We then use energy

$$\vec{\eta} \cdot \vec{H}_2(Y_t|Z) = \eta_F \log(O(Z, t)) Y_t + \eta_G \log(1 - O(Z, t))(1 - Y_t)$$

In the case of multiple objects or aspects of a single object, we use multiple templates and augment $Z$ with an indicator of the aspect $Z = (x, y, a)$. In our experiments on the dataset considered here, we found that the variability is too small (all horses facing left) to see a significant impact on performance from adding multiple aspects.

Lastly, we would like to capture the spatial layout of articulated structures such as the horses legs and head. To describe characteristic configuration of edges, we utilize the *geometric blur*[1] descriptor applied to the output of the $Pb$ boundary detector. The *geometric blur* centered at location $x$, $GB_x(y)$, is a linear operator applied to $Pb(x, y)$ whose value is another image given by the "convolution" of $Pb(x, y)$ with a spatially varying Gaussian. Geometric blur is motivated by the search for a linear operator which will respond strongly to a particular object feature and is invariant to some set of transformations of the image.

We use the geometric blur computed at the set of image edges ($Pb > 0.05$) to build a library of 64 prototypical "shapemes" from the training data by vector quantization. For each edge $X_e$ which expresses a particular shapeme we would like to know whether $X_e$ should be

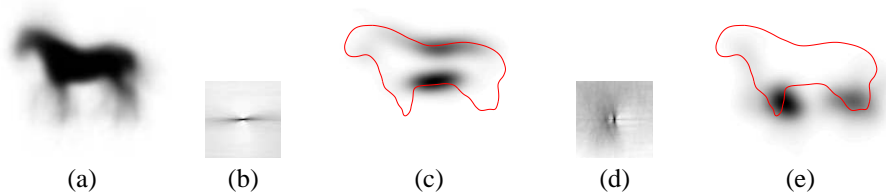

(a)        (b)        (c)        (d)        (e)

Figure 2: Using a priori shape knowledge: (a) average horse template. (b) one shapeme, capturing long horizontal curves. Shown here is the average shape in this shapeme cluster. (c) on a horse, this shapeme occurs at horse back and stomach. Shown here is the density of the shapeme $M^{ON}$ overlayed with a contour plot of the average mask. (d) another shapeme, capturing parallel vertical lines. (e) on a horse, this shapeme occurs at legs.

"turned on". This is estimated from training data by building spatial maps $M_i^{ON}(x,y)$ and $M_i^{OFF}(x,y)$ for each shapeme relative to the object center which record the frequency of a true/false boundary expressing shapeme $i$. Figure 3.3(b-e) shows two example shapemes and their corresponding $M^{ON}$ map. Let $S_{e,i}(x,y)$ be the indicator of the set of pixels on edge $e$ which express shapeme $i$. For an object in pose $Z = (x_0, y_0)$ we use the energy

$$\vec{\nu} \cdot \sum_e \vec{H}_3(X_e | Z, I) = \sum_e \frac{1}{|e|} (\nu_{ON} \sum_{i,x,y} \log(M_i^{ON}(x - x_0, y - y_0)) S_{e,i}(x,y) X_e +$$

$$\nu_{OFF} \sum_{i,x,y} \log(M_i^{OFF}(x - x_0, y - y_0)) S_{e,i}(x,y)(1 - X_e))$$

## 4    Learning cue integration

We carry out approximate inference using loopy belief propagation [15] which appears to converge quickly to a reasonable solution for the graphs and potentials in question.

To fit parameters of the model, we maximize the joint likelihood over $X, Y, Z$ taking each image as an iid sample. Since our model is log-linear in the parameters $\Theta$, partial derivatives always yield the difference between the empirical expectation of a feature given by the training data and the expected value given the model parameters. For example, the derivative with respect to the continuation parameter $\delta_0$ for a single training image/ground truth labeling, $(I, X, Y, Z)$ is:

$$\frac{\partial}{\partial \delta_0} - \log P(X, Y | Z, I, \Theta)$$

$$= \frac{\partial}{\partial \delta_0} \log \mathcal{Z}(I_n, \Theta) - \sum_V \frac{\partial}{\partial \delta_0} \{ \delta_0 \mathbf{1}_{\{\deg_g(V) + \deg_c(V) = 2\}} log(f(\theta)) \}$$

$$= \left\langle \sum_V \mathbf{1}_{\{deg_g(V) + \deg_c(V) = 2\}} log(f(\theta)) \right\rangle - \sum_V \mathbf{1}_{\{\deg_g(V) + \deg_c(V) = 2\}} log(f(\theta))$$

where the expectation is taken with respect to $P(X, Y | Z, I, \Theta)$.

Given this estimate, we optimize the parameters by gradient descent. We have also used the difference of the energy and the Bethe free energy given by the beliefs as an estimate of the log likelihood in order to support line-search in conjugate gradient or quasi-newton routines. For our model, we find that gradient descent with momentum is efficient enough.

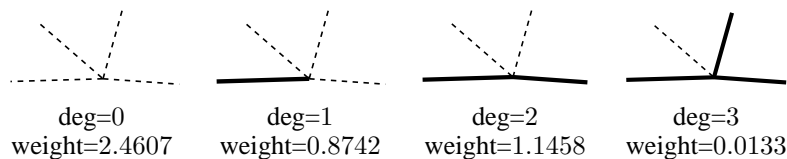

| deg=0 | deg=1 | deg=2 | deg=3 |
| weight=2.4607 | weight=0.8742 | weight=1.1458 | weight=0.0133 |

Figure 3: Learning about junctions: (a) deg=0, no boundary detected; the most common case. (b) line endings. (c) continuations of contours, more common than line endings. (d) T-junctions, very rare for the horse dataset. Compare with hand set potentials of Geman and Geman [3].

## 5 Experiments

In our experiments we use $344$ grayscale images of the horse dataset of Borenstein et al [2]. Half of the images are used for training and half for testing. Human-marked segmentations are used[1] for both training and evaluation.

**Training**: loopy belief propagation on a typical CDT graph converges in about $1$ second. The gradient descent learning described above converges within $1000$ iterations. To understand the weights given by the learning procedure, Figure 3 shows some of the junction types in $M_1$ and their associated weights $\delta$.

**Testing**: we evaluate the performance of our model on both edge and region labels. We present the results using a *precision-recall curve* which shows the trade-off between false positives and missed detections. For each edge $e$, we assign the marginal probability $E[X_e]$ to all pixels $(x, y)$ belonging to $e$. Then for each threshold $r$, pixels above $r$ are matched to human-marked boundaries $H$. The precision $P = P(H(x, y) = 1|P_E(x, y) > r)$ and recall $R = P(P_E(x, y) > r|H(x, y) = 1)$ are recorded. Similarly, each pixel in a triangle $t$ is assigned the marginal probability $E[Y_t]$ and the precision and recall of the ground-truth figural pixels computed.

The evaluations are shown in Figure 4 for various combinations of cues. Figure 5 shows our results on some of the test images.

## 6 Conclusion

We have introduced a conditional random field model on a triangulated representation of images for figure/ground labeling. We have measured the contributions of mid- and high-level cues by quantitative evaluations on held out test data. Our findings suggest that mid-level cues provide useful information, even in the presence of high-level shape cues. In future work we plan to extend this model to multiple object categories.

## Footnotes

[1]From the human segmentations on pixel-grid, we use two simple techniques to establish groundtruth labels on the CDT edges $X_e$ and triangles $Y_t$. For $X_e$, we run a maximum-cardinality bipartite matching between the human marked boundaries and the CDT edges. We label $X_e = 1$ if $75\%$ of the pixels lying under the edge $e$ are matched to human boundaries. For $Y_t$, we label $Y_t = 1$ if at least half of the pixels within the triangle are figural pixels in the human segmentation.

## References

[1] A. Berg and J. Malik. Geometric blur for template matching. In *CVPR*, 2001.

[2] E. Borenstein and S. Ullman. Class-specific, top-down segmentation. In *Proc. 7th Europ. Conf. Comput. Vision*, volume 2, pages 109–124, 2002.

[3] S. Geman and D. Geman. Stochastic relaxation, gibbs distribution, and the bayesian retoration of images. *IEEE Trans. Pattern Analysis and Machine Intelligence*, 6:721–41, Nov. 1984.

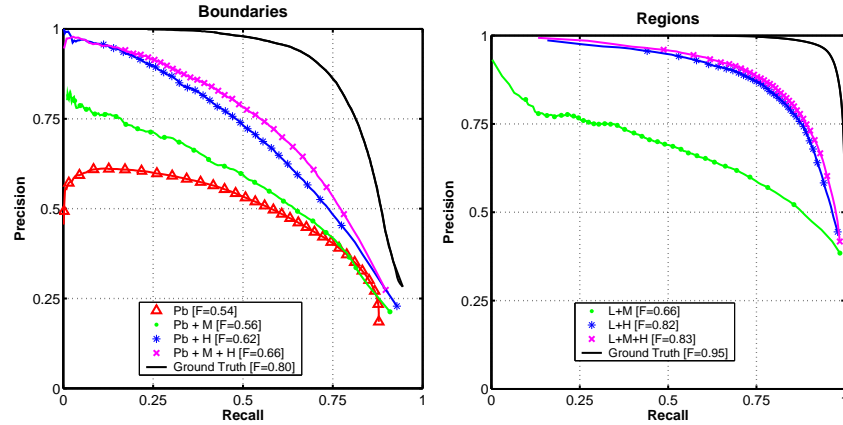

Figure 4: Performance evaluation: (a) precision-recall curves for horse boundaries, models with low-level cues only ($Pb$), low- plus mid-level cues ($Pb+M$), low- plus high-level cues ($Pb+H$), and all three classes of cues combined ($Pb+M+H$). The *F-measure* recorded in the legend is the maximal harmonic mean of precision and recall and provides an overall ranking. Using high-level cues greatly improves the boundary detection performance. Mid-level continuity cues are useful with or without high-level cues. (b) precision-recall for regions. The poor performance of the baseline $L+M$ model indicates the ambiguity of figure/ground labeling at low-level despite successful boundary detection. High-level shape knowledge is the key, consistent with evidence from psychophysics [10]. In both boundary and region cases, the groundtruth labels on CDTs are nearly perfect, indicating that the CDT graphs preserve most of the image structure.

[4] X. He, R. Zemel, and M. Carreira-Perpinan. Multiscale conditional random fields for image labelling. In *IEEE Conference on Computer Vision and Pattern Recognition*, 2004.

[5] M. P. Kumar, P. H. S. Torr, and A. Zisserman. OBJ CUT. In *CVPR*, 2005.

[6] S. Kumar and M. Hebert. Discriminative random fields: A discriminative framework for contextual interaction in classification. In *ICCV*, 2003.

[7] John Lafferty, Andrew McCallum, and Fernando Pereira. Conditional random fields: Probabilistic models for segmenting and labeling sequence data. In *Proc. 18th International Conf. on Machine Learning*, 2001.

[8] J. Malik, S. Belongie, J. Shi, and T. Leung. Textons, contours and regions: Cue integration in image segmentation. In *Proc. 7th Int'l. Conf. Computer Vision*, pages 918–925, 1999.

[9] D. Martin, C. Fowlkes, and J. Malik. Learning to detect natural image boundaries using brightness and texture. In *Advances in Neural Information Processing Systems 15*, 2002.

[10] M. A. Peterson and B. S. Gibson. Object recognition contributions to figure-ground organization. *Perception and Psychophysics*, 56:551–564, 1994.

[11] X. Ren, C. Fowlkes, and J. Malik. Mid-level cues improve boundary detection. Technical Report UCB//CSD-05-1382, UC Berkeley, January 2005.

[12] N. Shental, A. Zomet, T. Hertz, and Y. Weiss. Pairwise clustering and graphical models. In *NIPS 2003*, 2003.

[13] J. Shewchuk. Triangle: Engineering a 2d quality mesh generator and delaunay triangulator. In *First Workshop on Applied Computational Geometry*, pages 124–133, 1996.

[14] Z.W. Tu, X.R. Chen, A.L Yuille, and S.C. Zhu. Image parsing: segmentation, detection, and recognition. In *ICCV*, 2003.

[15] Y. Weiss. Correctness of local probability propagation in graphical models with loops. *Neural Computation*, 2000.

[16] S. Yu, R. Gross, and J. Shi. Concurrent object segmentation and recognition with graph partitioning. In *Advances in Neural Information Processing Systems 15*, 2002.

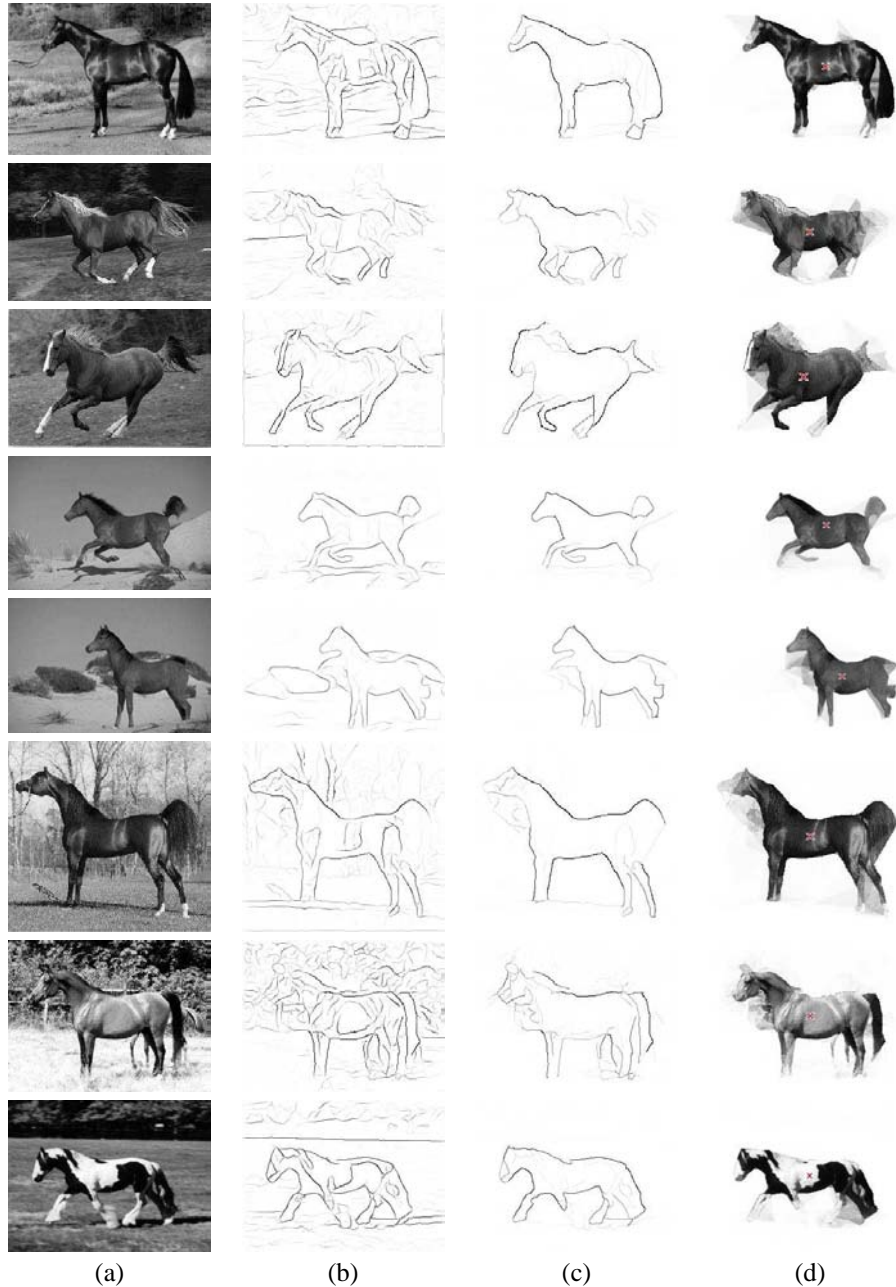

| (a) | (b) | (c) | (d) |

Figure 5: Sample results. (a) the input grayscale images. (b) the low-level boundary map output by $Pb$. (c) the edge marginals under our full model and (d) the image masked by the output region marginals. A red cross in (d) indicates the most probably object center. By combining relatively simple low-/mid-/high-level cues in a learning framework, We are able to find and segment horses under varying conditions with only a simple object mode. The boundary maps show the model is capable of suppressing strong gradients in the scene background while boosting low-contrast edges between figure and ground. (Row 3) shows an example of an unusual pose. In (Row 5) we predict a correct off-center object location and (Row 8) demonstrates grouping together figure with non-homogeneous appearance.